# Fused sparsity and robust estimation for linear models with unknown variance

**Yin Chen**
University Paris Est, LIGM
77455 Marne-la-Valle, FRANCE
yin.chen@eleves.enpc.fr

**Arnak S. Dalalyan**
ENSAE-CREST-GENES
92245 MALAKOFF Cedex, FRANCE
arnak.dalalyan@ensae.fr

## Abstract

In this paper, we develop a novel approach to the problem of learning sparse representations in the context of fused sparsity and unknown noise level. We propose an algorithm, termed Scaled Fused Dantzig Selector (SFDS), that accomplishes the aforementioned learning task by means of a second-order cone program. A special emphasize is put on the particular instance of fused sparsity corresponding to the learning in presence of outliers. We establish finite sample risk bounds and carry out an experimental evaluation on both synthetic and real data.

## 1 Introduction

Consider the classical problem of Gaussian linear regression[1]:

$$\boldsymbol{Y} = \mathbf{X}\boldsymbol{\beta}^* + \sigma^*\boldsymbol{\xi}, \qquad \boldsymbol{\xi} \sim \mathcal{N}_n(0, \mathbf{I}_n), \tag{1}$$

where $\boldsymbol{Y} \in \mathbb{R}^n$ and $\mathbf{X} \in \mathbb{R}^{n \times p}$ are observed, in the neoclassical setting of very large dimensional unknown vector $\boldsymbol{\beta}^*$. Even if the ambient dimensionality $p$ of $\boldsymbol{\beta}^*$ is larger than $n$, it has proven possible to consistently estimate this vector under the sparsity assumption. The letter states that the number of nonzero elements of $\boldsymbol{\beta}^*$, denoted by $s$ and called intrinsic dimension, is small compared to the sample size $n$. Most famous methods of estimating sparse vectors, the Lasso and the Dantzig Selector (DS), rely on convex relaxation of $\ell_0$-norm penalty leading to a convex program that involves the $\ell_1$-norm of $\boldsymbol{\beta}$. More precisely, for a given $\bar{\lambda} > 0$, the Lasso and the DS [26, 4, 5, 3] are defined as

$$\widehat{\boldsymbol{\beta}}^{\mathrm{L}} = \arg\min_{\boldsymbol{\beta} \in \mathbb{R}^p} \left\{ \frac{1}{2}\|\boldsymbol{Y} - \mathbf{X}\boldsymbol{\beta}\|_2^2 + \bar{\lambda}\|\boldsymbol{\beta}\|_1 \right\} \tag{Lasso}$$

$$\widehat{\boldsymbol{\beta}}^{\mathrm{DS}} = \arg\min \|\boldsymbol{\beta}\|_1 \text{ subject to } \|\mathbf{X}^\top(\boldsymbol{Y} - \mathbf{X}\boldsymbol{\beta})\|_\infty \leq \bar{\lambda}. \tag{DS}$$

The performance of these algorithms depends heavily on the choice of the tuning parameter $\bar{\lambda}$. Several empirical and theoretical studies emphasized that $\bar{\lambda}$ should be chosen proportionally to the noise standard deviation $\sigma^*$. Unfortunately, in most applications, the latter is unavailable. It is therefore vital to design statistical procedures that estimate $\boldsymbol{\beta}$ and $\sigma$ in a joint fashion. This topic received special attention in last years, cf. [10] and the references therein, with the introduction of computationally efficient and theoretically justified $\sigma$-adaptive procedures the square-root Lasso [2] (a.k.a. scaled Lasso [24]) and the $\ell_1$ penalized log-likelihood minimization [20].

In the present work, we are interested in the setting where $\boldsymbol{\beta}^*$ is not necessarily sparse, but for a known $q \times p$ matrix $\mathbf{M}$, the vector $\mathbf{M}\boldsymbol{\beta}^*$ is sparse. We call this setting "fused sparsity scenario".

The term "fused" sparsity, introduced by [27], originates from the case where $\mathbf{M}\boldsymbol{\beta}$ is the discrete derivative of a signal $\boldsymbol{\beta}$ and the aim is to minimize the total variation, see [12, 19] for a recent overview and some asymptotic results. For general matrices $\mathbf{M}$, tight risk bounds were proved in [14]. We adopt here this framework of general $\mathbf{M}$ and aim at designing a computationally efficient procedure capable to handle the situation of unknown noise level and for which we are able to provide theoretical guarantees along with empirical evidence for its good performance.

This goal is attained by introducing a new procedure, termed Scaled Fused Dantzig Selector (SFDS), which is closely related to the penalized maximum likelihood estimator but has some advantages in terms of computational complexity. We establish tight risk bounds for the SFDS, which are nearly as strong as those proved for the Lasso and the Dantzig selector in the case of known $\sigma^*$. We also show that the robust estimation in linear models can be seen as a particular example of the fused sparsity scenario. Finally, we carry out a "proof of concept" type experimental evaluation to show the potential of our approach.

## 2 Estimation under fused sparsity with unknown level of noise

### 2.1 Scaled Fused Dantzig Selector

We will only consider the case $\text{rank}(\mathbf{M}) = q \leq p$, which is more relevant for the applications we have in mind (image denoising and robust estimation). Under this condition, one can find a $(p-q) \times p$ matrix $\mathbf{N}$ such that the augmented matrix $\overline{\mathbf{M}} = [\mathbf{M}^\top \ \mathbf{N}^\top]^\top$ is of full rank. Let us denote by $\boldsymbol{m}_j$ the $j$th column of the matrix $\overline{\mathbf{M}}^{-1}$, so that $\overline{\mathbf{M}}^{-1} = [\boldsymbol{m}_1, ..., \boldsymbol{m}_p]$. We also introduce:

$$\overline{\mathbf{M}}^{-1} = [\mathbf{M}_\dagger, \mathbf{N}_\dagger], \quad \mathbf{M}_\dagger = [\boldsymbol{m}_1, ..., \boldsymbol{m}_q] \in \mathbb{R}^{p \times q}, \quad \mathbf{N}_\dagger = [\boldsymbol{m}_{q+1}, ..., \boldsymbol{m}_p] \in \mathbb{R}^{p \times (p-q)}.$$

Given two positive tuning parameters $\lambda$ and $\mu$, we define the Scaled Fused Dantzig Selector (SFDS) $(\widehat{\boldsymbol{\beta}}, \widehat{\sigma})$ as a solution to the following optimization problem:

$$\boxed{\text{minimize} \sum_{j=1}^{q} \|\mathbf{X}\boldsymbol{m}_j\|_2 |(\mathbf{M}\boldsymbol{\beta})_j| \text{ subject to } \begin{cases} |\boldsymbol{m}_j^\top \mathbf{X}^\top (\mathbf{X}\boldsymbol{\beta} - \boldsymbol{Y})| \leq \lambda\sigma \|\mathbf{X}\boldsymbol{m}_j\|_2, \ j \leq q; \\ \mathbf{N}_\dagger^\top \mathbf{X}^\top (\mathbf{X}\boldsymbol{\beta} - \boldsymbol{Y}) = 0, \\ n\mu\sigma^2 + \boldsymbol{Y}^\top \mathbf{X}\boldsymbol{\beta} \leq \|\boldsymbol{Y}\|_2^2. \end{cases} \quad \textbf{(P1)}}$$

This estimator has several attractive properties: (a) it can be efficiently computed even for very large scale problems using a second-order cone program, (b) it is equivariant with respect to the scale transformations both in the response $\boldsymbol{Y}$ and in the lines of $\mathbf{M}$ and, finally, (c) it is closely related to the penalized maximum likelihood estimator. Let us give further details on these points.

### 2.2 Relation with the penalized maximum likelihood estimator

One natural way to approach the problem of estimating $\boldsymbol{\beta}^*$ in our setup is to rely on the standard procedure of penalized log-likelihood minimization. If the noise distribution is Gaussian, $\boldsymbol{\xi} \sim \mathcal{N}_n(0, \mathbf{I}_n)$, the negative log-likelihood (up to irrelevant additive terms) is given by

$$\ell(\boldsymbol{Y}, \mathbf{X}; \boldsymbol{\beta}, \sigma) = n\log(\sigma) + \frac{\|\boldsymbol{Y} - \mathbf{X}\boldsymbol{\beta}\|_2^2}{2\sigma^2}.$$

In the context of large dimension we are concerned with, *i.e.*, when $p/n$ is not small, the maximum likelihood estimator is subject to overfitting and is of very poor quality. If it is plausible to expect that the data can be fitted sufficiently well by a vector $\boldsymbol{\beta}^*$ such that for some matrix $\mathbf{M}$, only a small fraction of elements of $\mathbf{M}\boldsymbol{\beta}^*$ are nonzero, then one can considerably improve the quality of estimation by adding a penalty term to the log-likelihood. However, the most appealing penalty, the number of nonzero elements of $\mathbf{M}\boldsymbol{\beta}$, leads to a nonconvex optimization problem which cannot be efficiently solved even for moderately large values of $p$. Instead, convex penalties of the form $\sum_j \omega_j |(\mathbf{M}\boldsymbol{\beta})_j|$, where $w_j > 0$ are some weights, have proven to provide high accuracy estimates at a relatively low computational cost. This corresponds to defining the estimator $(\widehat{\boldsymbol{\beta}}^{\text{PL}}, \widehat{\sigma}^{\text{PL}})$ as the

minimizer of the penalized log-likelihood

$$\bar{\ell}(\boldsymbol{Y}, \mathbf{X}; \boldsymbol{\beta}, \sigma) = n\log(\sigma) + \frac{\|\boldsymbol{Y} - \mathbf{X}\boldsymbol{\beta}\|_2^2}{2\sigma^2} + \sum_{j=1}^{q} \omega_j |(\mathbf{M}\boldsymbol{\beta})_j|.$$

To ensure the scale equivariance, the weights $\omega_j$ should be chosen inversely proportionally to $\sigma$: $\omega_j = \sigma^{-1}\bar{\omega}_j$. This leads to the estimator

$$(\widehat{\boldsymbol{\beta}}^{\mathrm{PL}}, \widehat{\sigma}^{\mathrm{PL}}) = \arg\min_{\boldsymbol{\beta}, \sigma} \left\{ n\log(\sigma) + \frac{\|\boldsymbol{Y} - \mathbf{X}\boldsymbol{\beta}\|_2^2}{2\sigma^2} + \sum_{j=1}^{q} \bar{\omega}_j \frac{|(\mathbf{M}\boldsymbol{\beta})_j|}{\sigma} \right\}.$$

Although this problem can be cast [20] as a problem of convex minimization (by making the change of parameters $\phi = \boldsymbol{\beta}/\sigma$ and $\rho = 1/\sigma$), it does not belong to the standard categories of convex problems that can be solved either by linear programming or by second-order cone programming or by semidefinite programming. Furthermore, the smooth part of the objective function is not Lipschitz which makes it impossible to directly apply most first-order optimization methods developed in recent years. Our goal is to propose a procedure that is close in spirit to the penalized maximum likelihood but has the additional property of being computable by standard algorithms of second-order cone programming.

To achieve this goal, at the first step, we remark that it can be useful to introduce a penalty term that depends exclusively on $\sigma$ and that prevents the estimator of $\sigma^*$ from being too large or too small. One can show that the only function (up to a multiplicative constant) that can serve as penalty without breaking the property of scale equivariance is the logarithmic function. Therefore, we introduce an additional tuning parameter $\mu > 0$ and look for minimizing the criterion

$$n\mu\log(\sigma) + \frac{\|\boldsymbol{Y} - \mathbf{X}\boldsymbol{\beta}\|_2^2}{2\sigma^2} + \sum_{j=1}^{q} \bar{\omega}_j \frac{|(\mathbf{M}\boldsymbol{\beta})_j|}{\sigma}. \tag{2}$$

If we make the change of variables $\phi_1 = \mathbf{M}\boldsymbol{\beta}/\sigma$, $\phi_2 = \mathbf{N}\boldsymbol{\beta}/\sigma$ and $\rho = 1/\sigma$, we get a convex function for which the first-order conditions [20] take the form

$$\boldsymbol{m}_j^\top \mathbf{X}^\top (\boldsymbol{Y} - \mathbf{X}\boldsymbol{\beta}) \in \bar{\omega}_j \mathrm{sign}(\{\mathbf{M}\boldsymbol{\beta}\}_j), \tag{3}$$

$$\mathbf{N}_\dagger^\top \mathbf{X}^\top (\boldsymbol{Y} - \mathbf{X}\boldsymbol{\beta}) = 0, \tag{4}$$

$$\frac{1}{n\mu}\left(\|\boldsymbol{Y}\|_2^2 - \boldsymbol{Y}^\top \mathbf{X}\boldsymbol{\beta}\right) = \sigma^2. \tag{5}$$

Thus, any minimizer of (2) should satisfy these conditions. Therefore, to simplify the problem of optimization we propose to replace minimization of (2) by the minimization of the weighted $\ell_1$-norm $\sum_j \bar{\omega}_j |(\mathbf{M}\boldsymbol{\beta})_j|$ subject to some constraints that are as close as possible to (3-5). The only problem here is that the constraints (3) and (5) are not convex. The "convexification" of these constraints leads to the procedure described in **(P1)**. As we explain below, the particular choice of $\bar{\omega}_j$s is dictated by the desire to enforce the scale equivariance of the procedure.

## 2.3 Basic properties

A key feature of the SFDS is its scale equivariance. Indeed, one easily checks that if $(\widehat{\boldsymbol{\beta}}, \widehat{\sigma})$ is a solution to **(P1)** for some inputs $\mathbf{X}$, $\boldsymbol{Y}$ and $\mathbf{M}$, then $\alpha(\widehat{\boldsymbol{\beta}}, \widehat{\sigma})$ will be a solution to **(P1)** for the inputs $\mathbf{X}$, $\alpha\boldsymbol{Y}$ and $\mathbf{M}$, whatever the value of $\alpha \in \mathbb{R}$ is. This is the equivariance with respect to the scale change in the response $\boldsymbol{Y}$. Our method is also equivariant with respect to the scale change in $\mathbf{M}$. More precisely, if $(\widehat{\boldsymbol{\beta}}, \widehat{\sigma})$ is a solution to **(P1)** for some inputs $\mathbf{X}$, $\boldsymbol{Y}$ and $\mathbf{M}$, then $(\widehat{\boldsymbol{\beta}}, \widehat{\sigma})$ will be a solution to **(P1)** for the inputs $\mathbf{X}$, $\boldsymbol{Y}$ and $\mathbf{DM}$, whatever the $q \times q$ diagonal matrix $\mathbf{D}$ is. The latter property is important since if we believe that for a given matrix $\mathbf{M}$ the vector $\mathbf{M}\boldsymbol{\beta}^*$ is sparse, then this is also the case for the vector $\mathbf{DM}\boldsymbol{\beta}^*$, for any diagonal matrix $\mathbf{D}$. Having a procedure the output of which is independent of the choice of $\mathbf{D}$ is of significant practical importance, since it leads to a solution that is robust with respect to small variations of the problem formulation.

The second attractive feature of the SFDS is that it can be computed by solving a convex optimization problem of second-order cone programming (SOCP). Recall that an SOCP is a constrained

optimization problem that can be cast as minimization with respect to $\boldsymbol{w} \in \mathbb{R}^d$ of a linear function $\boldsymbol{a}^\top \boldsymbol{w}$ under second-order conic constraints of the form $\|\mathbf{A}_i \boldsymbol{w} + \boldsymbol{b}_i\|_2 \leq \boldsymbol{c}_i^\top \boldsymbol{w} + d_i$, where $\mathbf{A}_i$s are some $r_i \times d$ matrices, $\boldsymbol{b}_i \in \mathbb{R}^{r_i}$, $\boldsymbol{c}_i \in \mathbb{R}^d$ are some vectors and $d_i$s are some real numbers. The problem **(P1)** belongs well to this category, since it can be written as $\min(u_1 + \ldots + u_q)$ subject to

$$\|\mathbf{X}\boldsymbol{m}_j\|_2 |(\mathbf{M}\boldsymbol{\beta})_j| \leq u_j; \qquad |\boldsymbol{m}_j^\top \mathbf{X}^\top (\mathbf{X}\boldsymbol{\beta} - \boldsymbol{Y})| \leq \lambda \sigma \|\mathbf{X}\boldsymbol{m}_j\|_2, \qquad \forall j = 1, \ldots, q;$$

$$\mathbf{N}_\dagger^\top \mathbf{X}^\top (\mathbf{X}\boldsymbol{\beta} - \boldsymbol{Y}) = 0, \qquad \sqrt{4n\mu\|\boldsymbol{Y}\|_2^2 \sigma^2 + (\boldsymbol{Y}^\top \mathbf{X}\boldsymbol{\beta})^2} \leq 2\|\boldsymbol{Y}\|_2^2 - \boldsymbol{Y}^\top \mathbf{X}\boldsymbol{\beta}.$$

Note that all these constraints can be transformed into linear inequalities, except the last one which is a second order cone constraint. The problems of this type can be efficiently solved by various standard toolboxes such as SeDuMi [22] or TFOCS [1].

## 2.4 Finite sample risk bound

To provide theoretical guarantees for our estimator, we impose the by now usual assumption of restricted eigenvalues on a suitably chosen matrix. This assumption, stated in Definition 2.1 below, was introduced and thoroughly discussed by [3]; we also refer the interested reader to [28].

**Définition** 2.1. *We say that a $n \times q$ matrix $\mathbf{A}$ satisfies the restricted eigenvalue condition* $\mathrm{RE}(s, 1)$, *if*

$$\kappa(s, 1) \triangleq \min_{|J| \leq s} \min_{\|\delta_{J^c}\|_1 \leq \|\delta_J\|_1} \frac{\|\mathbf{A}\delta\|_2}{\sqrt{n}\|\delta_J\|_2} > 0.$$

*We say that $\mathbf{A}$ satisfies the strong restricted eigenvalue condition* $\mathrm{RE}(s, s, 1)$, *if*

$$\kappa(s, s, 1) \triangleq \min_{|J| \leq s} \min_{\|\delta_{J^c}\|_1 \leq \|\delta_J\|_1} \frac{\|\mathbf{A}\delta\|_2}{\sqrt{n}\|\delta_{J \cup J_0}\|_2} > 0,$$

*where $J_0$ is the subset of $\{1, ..., q\}$ corresponding to the $s$ largest in absolute value coordinates of $\delta$.*

For notational convenience, we assume that $\mathbf{M}$ is normalized in such a way that the diagonal elements of $\frac{1}{n}\mathbf{M}_\dagger^\top \mathbf{X}^\top \mathbf{X} \mathbf{M}_\dagger$ are all equal to 1. This can always be done by multiplying $\mathbf{M}$ from the left by a suitably chosen positive definite diagonal matrix. Furthermore, we will repeatedly use the projector[2] $\boldsymbol{\Pi} = \mathbf{X}\mathbf{N}_\dagger (\mathbf{N}_\dagger^\top \mathbf{X}^\top \mathbf{X} \mathbf{N}_\dagger)^{-1} \mathbf{N}_\dagger^\top \mathbf{X}^\top$ onto the subspace of $\mathbb{R}^n$ spanned by the columns of $\mathbf{X}\mathbf{N}_\dagger$. We denote by $r = \mathrm{rank}\{\boldsymbol{\Pi}\}$ the rank of this projector which is typically very small compared to $n \wedge p$, and is always smaller than $n \wedge (p - q)$. In all theoretical results, the matrices $\mathbf{X}$ and $\mathbf{M}$ are assumed deterministic.

**Theorem 2.1.** *Let us fix a tolerance level $\delta \in (0, 1)$ and define $\lambda = \sqrt{2n\gamma \log(q/\delta)}$. Assume that the tuning parameters $\gamma, \mu > 0$ satisfy*

$$\frac{\mu}{\gamma} \leq 1 - \frac{r}{n} - 2\frac{\sqrt{(n-r)\log(1/\delta)} + \log(1/\delta)}{n}. \tag{6}$$

*If the vector $\mathbf{M}\boldsymbol{\beta}^*$ is $s$-sparse and the matrix $(\mathbf{I}_n - \boldsymbol{\Pi})\mathbf{X}\mathbf{M}_\dagger$ satisfies the condition $\mathrm{RE}(s, 1)$ with some $\kappa > 0$ then, with probability at least $1 - 6\delta$, it holds:*

$$\|\mathbf{M}(\widehat{\boldsymbol{\beta}} - \boldsymbol{\beta}^*)\|_1 \leq \frac{4}{\kappa^2}(\widehat{\sigma} + \sigma^*)s\sqrt{\frac{2\gamma \log(q/\delta)}{n}} + \frac{\sigma^*}{\kappa}\sqrt{\frac{2s \log(1/\delta)}{n}} \tag{7}$$

$$\|\mathbf{X}(\widehat{\boldsymbol{\beta}} - \boldsymbol{\beta}^*)\|_2 \leq 2(\widehat{\sigma} + \sigma^*)\frac{\sqrt{2\gamma s \log(q/\delta)}}{\kappa} + \sigma^*\left(\sqrt{8 \log(1/\delta)} + r\right). \tag{8}$$

*If, in addition, $(\mathbf{I}_n - \boldsymbol{\Pi})\mathbf{X}\mathbf{M}_\dagger$ satisfies the condition $RE(s, s, 1)$ with some $\kappa > 0$ then, with a probability at least $1 - 6\delta$, we have:*

$$\|\mathbf{M}\widehat{\boldsymbol{\beta}} - \mathbf{M}\boldsymbol{\beta}^*\|_2 \leq \frac{4(\widehat{\sigma} + \sigma^*)}{\kappa^2}\sqrt{\frac{2s \log(q/\delta)}{n}} + \frac{\sigma^*}{\kappa}\sqrt{\frac{2 \log(1/\delta)}{n}} \tag{9}$$

*Moreover, with a probability at least $1 - 7\delta$, we have:*

$$\widehat{\sigma} \leq \frac{\sigma^*}{\mu^{1/2}} + \frac{\lambda\|\mathbf{M}\boldsymbol{\beta}^*\|_1}{n\mu} + \frac{s^{1/2}\sigma^* \log(q/\delta)}{n\kappa\mu^{1/2}} + (\sigma^* + \|\mathbf{M}\boldsymbol{\beta}^*\|_1)\mu^{-1/2}\sqrt{\frac{2 \log(1/\delta)}{n}}. \tag{10}$$

Before looking at the consequences of these risk bounds in the particular case of robust estimation, let us present some comments highlighting the claims of Theorem 2.1. The first comment is about the conditions on the tuning parameters $\mu$ and $\gamma$. It is interesting to observe that the roles of these parameters are very clearly defined: $\gamma$ controls the quality of estimating $\boldsymbol{\beta}^*$ while $\mu$ determines the quality of estimating $\sigma^*$. One can note that all the quantities entering in the right-hand side of (6) are known, so that it is not hard to choose $\mu$ and $\gamma$ in such a way that they satisfy the conditions of Theorem 2.1. However, in practice, this theoretical choice may be too conservative in which case it could be a better idea to rely on cross validation.

The second remark is about the rates of convergence. According to (8), the rate of estimation measured in the mean prediction loss $\frac{1}{n}\|\mathbf{X}(\widehat{\boldsymbol{\beta}} - \boldsymbol{\beta}^*)\|_2^2$ is of the order of $s\log(q)/n$, which is known as fast or parametric rate. The vector $\mathbf{M}\boldsymbol{\beta}^*$ is also estimated with the nearly parametric rate in both $\ell_1$ and $\ell_2$-norms. To the best of our knowledge, this is the first work where such kind of fast rates are derived in the context of fused sparsity with unknown noise-level. With some extra work, one can check that if, for instance, $\gamma = 1$ and $|\mu - 1| \leq cn^{-1/2}$ for some constant $c$, then the estimator $\widehat{\sigma}$ has also a risk of the order of $sn^{-1/2}$. However, the price to pay for being adaptive with respect to the noise level is the presence of $\|\mathbf{M}\boldsymbol{\beta}^*\|_1$ in the bound on $\widehat{\sigma}$, which deteriorates the quality of estimation in the case of large signal-to-noise ratio.

Even if Theorem 2.1 requires the noise distribution to be Gaussian, the proposed algorithm remains valid in a far broader context and tight risk bounds can be obtained under more general conditions on the noise distribution. In fact, one can see from the proof that we only need to know confidence sets for some linear and quadratic functionals of $\boldsymbol{\xi}$. For instance, such kind of confidence sets can be readily obtained in the case of bounded errors $\xi_i$ using the Bernstein inequality. It is also worthwhile to mention that the proof of Theorem 2.1 is not a simple adaptation of the arguments used to prove analogous results for ordinary sparsity, but contains some qualitatively novel ideas. More precisely, the cornerstone of the proof of risk bounds for the Dantzig selector [4, 3, 9] is that the true parameter $\boldsymbol{\beta}^*$ is a feasible solution. In our case, this argument cannot be used anymore. Our proposal is then to specify another vector $\widetilde{\boldsymbol{\beta}}$ that simultaneously satisfies the following three conditions: $\mathbf{M}\widetilde{\boldsymbol{\beta}}$ has the same sparsity pattern as $\mathbf{M}\boldsymbol{\beta}^*$, $\widetilde{\boldsymbol{\beta}}$ is close to $\boldsymbol{\beta}^*$ and lies in the feasible set.

A last remark is about the restricted eigenvalue conditions. They are somewhat cumbersome in this abstract setting, but simplify a lot when the concrete example of robust estimation is considered, cf. the next section. At a heuristical level, these conditions require from the columns of $\mathbf{X}\mathbf{M}_\dagger$ to be not very strongly correlated. Unfortunately, this condition fails for the matrices appearing in the problem of multiple change-point detection, which is an important particular instance of fused sparsity. There are some workarounds to circumvent this limitation in that particular setting, see [17, 11]. The extension of these kind of arguments to the case of unknown $\sigma^*$ is an open problem we intend to tackle in the near future.

## 3 Application to robust estimation

This methodology can be applied in the context of robust estimation, *i.e.*, when we observe $\boldsymbol{Y} \in \mathbb{R}^n$ and $\mathbf{A} \in \mathbb{R}^{n \times k}$ such that the relation

$$Y_i = (\mathbf{A}\boldsymbol{\theta}^*)_i + \sigma^*\xi_i, \qquad \xi_i \overset{\text{iid}}{\sim} \mathcal{N}(0, 1)$$

holds only for some indexes $i \in \mathcal{I} \subset \{1, ..., n\}$, called inliers. The indexes does not belonging to $\mathcal{I}$ will be referred to as outliers. The setting we are interested in is the one frequently encountered in computer vision [13, 25]: the dimensionality $k$ of $\boldsymbol{\theta}^*$ is small as compared to $n$ but the presence of outliers causes the complete failure of the least squares estimator. In what follows, we use the standard assumption that the matrix $\frac{1}{n}\mathbf{A}^\top\mathbf{A}$ has diagonal entries equal to one.

Following the ideas developed in [6, 7, 8, 18, 15], we introduce a new vector $\boldsymbol{\omega} \in \mathbb{R}^n$ that serves to characterize the outliers. If an entry $\omega_i$ of $\boldsymbol{\omega}$ is nonzero, then the corresponding observation $Y_i$ is an outlier. This leads to the model:

$$\boldsymbol{Y} = \mathbf{A}\boldsymbol{\theta}^* + \sqrt{n}\boldsymbol{\omega}^* + \sigma^*\boldsymbol{\xi} = \mathbf{X}\boldsymbol{\beta}^* + \sigma^*\boldsymbol{\xi}, \quad \text{where} \quad \mathbf{X} = [\sqrt{n}\,\mathbf{I}_n\ \mathbf{A}], \quad \text{and} \quad \boldsymbol{\beta} = [\boldsymbol{\omega}^*\,; \boldsymbol{\theta}^*]^\top.$$

Thus, we have rewritten the problem of robust estimation in linear models as a problem of estimation in high dimension under the fused sparsity scenario. Indeed, we have $\mathbf{X} \in \mathbb{R}^{n \times (n+k)}$

and $\boldsymbol{\beta}^* \in \mathbb{R}^{n+k}$, and we are interested in finding an estimator $\widehat{\boldsymbol{\beta}}$ of $\boldsymbol{\beta}^*$ for which $\widehat{\boldsymbol{\omega}} = [\mathbf{I}_n \mathbf{0}_{n \times k}]\widehat{\boldsymbol{\beta}}$ contains as many zeros as possible. This means that we expect that the number of outliers is significantly smaller than the sample size. We are thus in the setting of fused sparsity with $\mathbf{M} = [\mathbf{I}_n\ \mathbf{0}_{n \times k}]$. Setting $\mathbf{N} = [\mathbf{0}_{k \times n}\ \mathbf{I}_k]$, we define the Scaled Robust Dantzig Selector (SRDS) as a solution $(\widehat{\boldsymbol{\theta}}, \widehat{\boldsymbol{\omega}}, \widehat{\sigma})$ of the problem:

$$
\text{minimize } \|\boldsymbol{\omega}\|_1 \quad \text{subject to} \begin{cases} \sqrt{n}\|\mathbf{A}\boldsymbol{\theta} + \sqrt{n}\,\boldsymbol{\omega} - \boldsymbol{Y}\|_\infty \leq \lambda\sigma, \\ \mathbf{A}^\top(\mathbf{A}\boldsymbol{\theta} + \sqrt{n}\,\boldsymbol{\omega} - \boldsymbol{Y}) = 0, \\ n\mu\sigma^2 + \boldsymbol{Y}^\top(\mathbf{A}\boldsymbol{\theta} + \sqrt{n}\boldsymbol{\omega}) \leq \|\boldsymbol{Y}\|_2^2. \end{cases} \tag{P2}
$$

Once again, this can be recast in a SOCP and solved with great efficiency by standard algorithms. Furthermore, the results of the previous section provide us with strong theoretical guarantees for the SRDS. To state the corresponding result, we will need a notation for the largest and the smallest singular values of $\frac{1}{\sqrt{n}}\mathbf{A}$ denoted by $\nu^*$ and $\nu_*$ respectively.

**Theorem 3.1.** *Let us fix a tolerance level $\delta \in (0,1)$ and define $\lambda = \sqrt{2n\gamma \log(n/\delta)}$. Assume that the tuning parameters $\gamma, \mu > 0$ satisfy $\frac{\mu}{\gamma} \leq 1 - \frac{k}{n} - \frac{2}{n}\big(\sqrt{(n-k)\log(1/\delta)} + \log(1/\delta)\big)$. Let $\boldsymbol{\Pi}$ denote the orthogonal projector onto the $k$-dimensional subspace of $\mathbb{R}^n$ spanned by the columns of $\mathbf{A}$. If the vector $\boldsymbol{\omega}^*$ is $s$-sparse and the matrix $\sqrt{n}(\mathbf{I}_n - \boldsymbol{\Pi})$ satisfies the condition $\mathrm{RE}(s,1)$ with some $\kappa > 0$ then, with probability at least $1 - 5\delta$, it holds:*

$$
\|\widehat{\boldsymbol{\omega}} - \boldsymbol{\omega}^*\|_1 \leq \frac{4}{\kappa^2}(\widehat{\sigma} + \sigma^*)s\sqrt{\frac{2\gamma \log(n/\delta)}{n}} + \frac{\sigma^*}{\kappa}\sqrt{\frac{2s \log(1/\delta)}{n}}, \tag{11}
$$

$$
\|(\mathbf{I}_n - \boldsymbol{\Pi})(\widehat{\boldsymbol{\omega}} - \boldsymbol{\omega}^*)\|_2 \leq \frac{2(\widehat{\sigma} + \sigma^*)}{\kappa}\sqrt{\frac{2s \log(n/\delta)}{n}} + \sigma^*\sqrt{\frac{2 \log(1/\delta)}{n}}. \tag{12}
$$

*If, in addition, $\sqrt{n}\,(\mathbf{I}_n - \boldsymbol{\Pi})$ satisfies the condition $\mathrm{RE}(s,s,1)$ with some $\kappa > 0$ then, with a probability at least $1 - 6\delta$, we have:*

$$
\|\widehat{\boldsymbol{\omega}} - \boldsymbol{\omega}^*\|_2 \leq \frac{4(\widehat{\sigma} + \sigma^*)}{\kappa^2}\sqrt{\frac{2s \log(n/\delta)}{n}} + \frac{\sigma^*}{\kappa}\sqrt{\frac{2 \log(1/\delta)}{n}}
$$

$$
\|\widehat{\boldsymbol{\theta}} - \boldsymbol{\theta}^*\|_2 \leq \frac{\nu^*}{\nu_*^2}\left\{\frac{4(\widehat{\sigma} + \sigma^*)}{\kappa^2}\sqrt{\frac{2s \log(n/\delta)}{n}} + \frac{\sigma^*}{\kappa}\sqrt{\frac{2 \log(1/\delta)}{n}} + \frac{\sigma^*(\sqrt{k} + \sqrt{2\log(1/\delta)})}{\sqrt{n}}\right\}
$$

*Moreover, with a probability at least $1 - 7\delta$, the following inequality holds:*

$$
\widehat{\sigma} \leq \frac{\sigma^*}{\mu^{1/2}} + \frac{\lambda\|\boldsymbol{\omega}^*\|_1}{n\mu} + \frac{s^{1/2}\sigma^* \log(n/\delta)}{n\kappa\mu^{1/2}} + (\sigma^* + \|\boldsymbol{\omega}^*\|_1)\mu^{-1/2}\sqrt{\frac{2\log(1/\delta)}{n}}. \tag{13}
$$

All the comments made after Theorem 2.1, especially those concerning the tuning parameters and the rates of convergence, hold true for the risk bounds in Theorem 3.1 as well. Furthermore, the restricted eigenvalue condition in the latter theorem is much simpler and deserves a special attention. In particular, one can remark that the failure of $\mathrm{RE}(s,1)$ for $\sqrt{n}(\mathbf{I}_n - \boldsymbol{\Pi})$ implies that there is a unit vector $\boldsymbol{\delta}$ in $\mathrm{Im}(\mathbf{A})$ such that $|\delta_{(1)}| + \ldots + |\delta_{(n-s)}| \leq |\delta_{(n-s+1)}| + \ldots + |\delta_{(n)}|$, where $\delta_{(k)}$ stands for the $k$th smallest (in absolute value) entry of $\boldsymbol{\delta}$. To gain a better understanding of how restrictive this assumption is, let us consider the case where the rows $\boldsymbol{a}_1, \ldots, \boldsymbol{a}_n$ of $\mathbf{A}$ are i.i.d. zero mean Gaussian vectors. Since $\boldsymbol{\delta} \in \mathrm{Im}(\mathbf{A})$, its coordinates $\delta_i$ are also i.i.d. Gaussian random variables (they can be considered $\mathcal{N}(0,1)$ due to the homogeneity of the inequality we are interested in). The inequality $|\delta_{(1)}| + \ldots + |\delta_{(n-s)}| \leq |\delta_{(n-s+1)}| + \ldots + |\delta_{(n)}|$ can be written as $\frac{1}{n}\sum_i |\delta_i| \leq \frac{2}{n}(|\delta_{(n-s+1)}| + \ldots + |\delta_{(n)}|)$. While the left-hand side of this inequality tends to $\mathbb{E}[|\delta_1|] > 0$, the right-hand side is upper-bounded by $\frac{2s}{n}\max_i |\delta_i|$, which is on the order of $\frac{2s\sqrt{\log n}}{n}$. Therefore, if $\frac{2s\sqrt{\log n}}{n}$ is small, the condition $\mathrm{RE}(s,1)$ is satisfied. This informal discussion can be made rigorous by studying large deviations of the quantity $\max_{\boldsymbol{\delta} \in \mathrm{Im}(\mathbf{A})\setminus\{0\}} \|\boldsymbol{\delta}\|_\infty/\|\boldsymbol{\delta}\|_1$. A simple sufficient condition entailing $\mathrm{RE}(s,1)$ for $\sqrt{n}(\mathbf{I}_n - \boldsymbol{\Pi})$ is presented in the following lemma.

**Lemma 3.2.** *Let us set $\zeta_s(\mathbf{A}) = \inf_{\boldsymbol{u} \in \mathbb{S}^{k-1}} \frac{1}{n}\sum_{i=1}^n |\boldsymbol{a}_i\boldsymbol{u}| - \frac{2s\|\mathbf{A}\|_{2,\infty}}{\sqrt{n}}$. If $\zeta_s(\mathbf{A}) > 0$, then $\sqrt{n}\,(\mathbf{I}_n - \boldsymbol{\Pi})$ satisfies both $RE(s,1)$ and $RE(s,s,1)$ with $\kappa(s,1) \geq \kappa(s,s,1) \geq \zeta_s(\mathbf{A})/\sqrt{(\nu^*)^2 + \zeta_s(\mathbf{A})^2}$.*

| | SFDS | | | | Lasso | | Square-Root Lasso | | | |
|---|---|---|---|---|---|---|---|---|---|---|
| | $|\widehat{\boldsymbol{\beta}} - \boldsymbol{\beta}^*|_2$ | | $|\widehat{\sigma} - \sigma^*|$ | | $|\widehat{\boldsymbol{\beta}} - \boldsymbol{\beta}^*|_2$ | | $|\widehat{\boldsymbol{\beta}} - \boldsymbol{\beta}^*|_2$ | | $|\widehat{\sigma} - \sigma^*|$ | |
| $(T,\ p,\ s^*,\sigma^*)$ | Ave | StD | Ave | StD | Ave | StD | Ave | StD | Ave | StD |
| $(200,400,2,.5)$ | 0.04 | 0.03 | 0.18 | 0.14 | 0.07 | 0.05 | 0.06 | 0.04 | 0.20 | 0.14 |
| $(200,400,2,\ 1)$ | 0.09 | 0.05 | 0.42 | 0.35 | 0.16 | 0.11 | 0.13 | 0.09 | 0.46 | 0.37 |
| $(200,400,2,\ 2)$ | 0.23 | 0.17 | 0.75 | 0.55 | 0.31 | 0.21 | 0.25 | 0.18 | 0.79 | 0.56 |
| $(200,400,5,.5)$ | 0.06 | 0.01 | 0.28 | 0.11 | 0.13 | 0.09 | 0.11 | 0.06 | 0.18 | 0.27 |
| $(200,400,5,\ 1)$ | 0.20 | 0.05 | 0.56 | 0.10 | 0.31 | 0.04 | 0.25 | 0.02 | 0.66 | 0.05 |
| $(200,400,5,\ 2)$ | 0.34 | 0.11 | 0.34 | 0.21 | 0.73 | 0.25 | 0.47 | 0.29 | 0.69 | 0.70 |
| $(200,400,10,.5)$ | 0.10 | 0.01 | 0.36 | 0.02 | 0.15 | 0.00 | 0.10 | 0.01 | 0.36 | 0.02 |
| $(200,400,10,\ 1)$ | 0.19 | 0.09 | 0.27 | 0.26 | 0.31 | 0.04 | 0.19 | 0.09 | 0.27 | 0.26 |
| $(200,400,10,\ 2)$ | 1.90 | 0.20 | 4.74 | 1.01 | 0.61 | 0.08 | 1.80 | 0.04 | 3.70 | 0.48 |

Table 1: Comparing our procedure SFDS with the (oracle) Lasso and the SqRL on a synthetic dataset. The average values and the standard deviations of the quantities $|\widehat{\boldsymbol{\beta}} - \boldsymbol{\beta}^*|_2$ and $|\widehat{\sigma} - \sigma^*|$ over 500 trials are reported. They represent respectively the accuracy in estimating the regression vector and the level of noise.

The proof of the lemma can be found in the supplementary material.

One can take note that the problem **(P2)** boils down to computing $(\widehat{\boldsymbol{\omega}}, \widehat{\sigma})$ as a solution to

$$\text{minimize } \|\boldsymbol{\omega}\|_1 \quad \text{subject to } \begin{cases} \sqrt{n}\|(\mathbf{I}_n - \boldsymbol{\Pi})(\sqrt{n}\boldsymbol{\omega} - \boldsymbol{Y})\|_\infty \leq \lambda\sigma, \\ n\mu\sigma^2 + \sqrt{n}[(\mathbf{I}_n - \boldsymbol{\Pi})\boldsymbol{Y}]^\top \boldsymbol{\omega} \leq \|(\mathbf{I}_n - \boldsymbol{\Pi})\boldsymbol{Y}\|_2^2. \end{cases}$$

and then setting $\widehat{\boldsymbol{\theta}} = (\mathbf{A}^\top \mathbf{A})^{-1}\mathbf{A}^\top(\boldsymbol{Y} - \sqrt{n}\,\widehat{\boldsymbol{\omega}})$.

# 4 Experiments

For the empirical evaluation we use a synthetic dataset with randomly drawn Gaussian design matrix $\mathbf{X}$ and the real-world dataset fountain-P11[3], on which we apply our methodology for computing the fundamental matrices between consecutive images.

## 4.1 Comparative evaluation on synthetic data

We randomly generated a $n \times p$ matrix $\mathbf{X}$ with independent entries distributed according to the standard normal distribution. Then we chose a vector $\boldsymbol{\beta}^* \in \mathbb{R}^p$ that has exactly $s$ nonzero elements all equal to one. The indexes of these elements were chosen at random. Finally, the response $\boldsymbol{Y} \in \mathbb{R}^n$ was computed by adding a random noise $\sigma^* N_n(0, \mathbf{I}_n)$ to the signal $\mathbf{X}\boldsymbol{\beta}^*$. Once $\boldsymbol{Y}$ and $\mathbf{X}$ available, we computed three estimators of the parameters using the standard sparsity penalization (in order to be able to compare our approach to the others): the SFDS, the Lasso and the square-root Lasso (SqRL). We used the "universal" tuning parameters for all these methods: $(\lambda, \mu) = (\sqrt{2n\log(p)}, 1)$ for the SFDS, $\lambda = \sqrt{2\log(p)}$ for the SqRL and $\lambda = \sigma^*\sqrt{2\log(p)}$ for the Lasso. Note that the latter is not really an estimator but rather an oracle since it exploits the knowledge of the true $\sigma^*$. This is why the accuracy in estimating $\sigma^*$ is not reported in Table 1. To reduce the well known bias toward zero [4, 23], we performed a post-processing for all of three procedures. It consisted in computing least squares estimators after removing all the covariates corresponding to vanishing coefficients of the estimator of $\boldsymbol{\beta}^*$. The results summarized in Table 1 show that the SFDS is competitive with the state-of-the-art methods and, a bit surprisingly, is sometimes more accurate than the oracle Lasso using the true variance in the penalization. We stress however that the SFDS is designed for being applied in—and has theoretical guarantees for—the broader setting of fused sparsity.

## 4.2 Robust estimation of the fundamental matrix

To provide a qualitative evaluation of the proposed methodology on real data, we applied the SRDS to the problem of fundamental matrix estimation in multiple-view geometry, which constitutes an

|  | 1 | 2 | 3 | 4 | 5 | 6 | 7 | 8 | 9 | 10 | Average |
|---|---|---|---|---|---|---|---|---|---|---|---|
| $\widehat{\sigma}$ | 0.13 | 0.13 | 0.13 | 0.17 | 0.16 | 0.17 | 0.20 | 0.18 | 0.17 | 0.11 | 0.15 |
| $\|\widehat{\boldsymbol{\omega}}\|_0$ | 218 | 80 | 236 | 90 | 198 | 309 | 17 | 31 | 207 | 8 | 139.4 |
| $\frac{100}{n}\|\widehat{\boldsymbol{\omega}}\|_0$ | 1.3 | 0.46 | 1.37 | 0.52 | 1.13 | 1.84 | 0.12 | 0.19 | 1.49 | 1.02 | 0.94 |

Table 2: Quantitative results on fountain dataset.

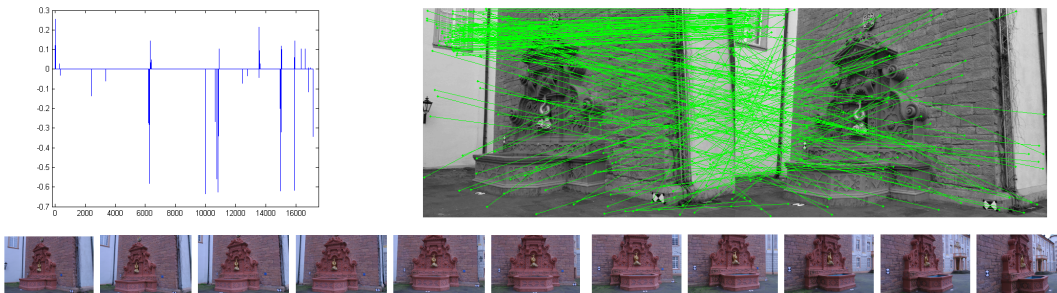

Figure 1: Qualitative results on fountain dataset. Top left: the values of $\widehat{\omega}_i$ for the first pair of images. There is a clear separation between outliers and inliers. Top right: the first pair of images and the matches classified as wrong by SRDS. Bottom: the eleven images of the dataset.

essential step in almost all pipelines of 3D reconstruction [13, 25]. In short, if we have two images $I$ and $I'$ representing the same 3D scene, then there is a $3\times3$ matrix $\mathsf{F}$, called fundamental matrix, such that a point $\mathbf{x} = (x, y)$ in $I_1$ matches with the point $\mathbf{x}' = (x', y')$ in $I'$ only if $[x; y; 1]\,\mathsf{F}\,[x'; y'; 1]^\top = 0$. Clearly, $\mathsf{F}$ is defined up to a scale factor: if $F_{33} \neq 0$, one can assume that $F_{33} = 1$. Thus, each pair $\mathbf{x} \leftrightarrow \mathbf{x}'$ of matching points in images $I$ and $I'$ yields a linear constraint on the eight remaining coefficients of $\mathsf{F}$. Because of the quantification and the presence of noise in images, these linear relations are satisfied up to some error. Thus, estimation of $\mathsf{F}$ from a family of matching points $\{\mathbf{x}_i \leftrightarrow \mathbf{x}'_i; i = 1, \ldots, n\}$ is a problem of linear regression. Typically, matches are computed by comparing local descriptors (such as SIFT [16]) and, for images of reasonable resolution, hundreds of matching points are found. The computation of the fundamental matrix would not be a problem in this context of large sample size / low dimension, if the matching algorithms were perfectly correct. However, due to noise, repetitive structures and other factors, a non-negligible fraction of detected matches are wrong (outliers). Elimination of these outliers and robust estimation of $\mathsf{F}$ are crucial steps for performing 3D reconstruction.

Here, we apply the SRDS to the problem of estimation of $\mathsf{F}$ for 10 pairs of consecutive images provided by the fountain dataset [21]: the 11 images are shown at the bottom of Fig. 1. Using SIFT descriptors, we found more than 17.000 point matches in most pairs of images among the 10 pairs we are considering. The CPU time for computing each matrix using the SeDuMi solver [22] was about 7 seconds, despite such a large dimensionality. The number of outliers and the estimated noise-level for each pair of images are reported in Table 2. We also showed in Fig. 1 the 218 outliers for the first pair of images. They are all indeed wrong correspondncies, even those which correspond to the windows (this is due to the repetitive structure of the window).

## 5   Conclusion and perspectives

We have presented a new procedure, SFDS, for the problem of learning linear models with unknown noise level under the fused sparsity scenario. We showed that this procedure is inspired by the penalized maximum likelihood but has the advantage of being computable by solving a second-order cone program. We established tight, nonasymptotic, theoretical guarantees for the SFDS with a special attention paid to robust estimation in linear models. The experiments we have carried out are very promising and support our theoretical results.

In the future, we intend to generalize the theoretical study of the performance of the SFDS to the case of non-Gaussian errors $\xi_i$, as well as to investigate its power in variable selection. The extension to the case where the number of lines in $\mathbf{M}$ is larger than the number of columns is another interesting topic for future research.

## Footnotes

[1]We denote by $\mathbf{I}_n$ the $n \times n$ identity matrix. For a vector $\boldsymbol{v}$, we use the standard notation $\|\boldsymbol{v}\|_1$, $\|\boldsymbol{v}\|_2$ and $\|\boldsymbol{v}\|_\infty$ for the $\ell_1$, $\ell_2$ and $\ell_\infty$ norms, corresponding respectively to the sum of absolute values, the square root of the sum of squares and the maximum of the coefficients of $\boldsymbol{v}$.

[2]Here and in the sequel, the inverse of a singular matrix is understood as MoorePenrose pseudoinverse.

[3]available at `http://cvlab.epfl.ch/~strecha/multiview/denseMVS.html`

# References

[1] Stephen Becker, Emmanuel Candès, and Michael Grant. Templates for convex cone problems with applications to sparse signal recovery. *Math. Program. Comput.*, 3(3):165–218, 2011.

[2] A. Belloni, Victor Chernozhukov, and L. Wang. Square-root lasso: Pivotal recovery of sparse signals via conic programming. *Biometrika*, to appear, 2012.

[3] Peter J. Bickel, Ya'acov Ritov, and Alexandre B. Tsybakov. Simultaneous analysis of lasso and Dantzig selector. *Ann. Statist.*, 37(4):1705–1732, 2009.

[4] Emmanuel Candes and Terence Tao. The Dantzig selector: statistical estimation when $p$ is much larger than $n$. *Ann. Statist.*, 35(6):2313–2351, 2007.

[5] Emmanuel J. Candès. The restricted isometry property and its implications for compressed sensing. *C. R. Math. Acad. Sci. Paris*, 346(9-10):589–592, 2008.

[6] Emmanuel J. Candès and Paige A. Randall. Highly robust error correction by convex programming. *IEEE Trans. Inform. Theory*, 54(7):2829–2840, 2008.

[7] Arnak S. Dalalyan and Renaud Keriven. $L_1$-penalized robust estimation for a class of inverse problems arising in multiview geometry. In *NIPS*, pages 441–449, 2009.

[8] Arnak S. Dalalyan and Renaud Keriven. Robust estimation for an inverse problem arising in multiview geometry. *J. Math. Imaging Vision.*, 43(1):10–23, 2012.

[9] Eric Gautier and Alexandre Tsybakov. High-dimensional instrumental variables regression and confidence sets. Technical Report arxiv:1105.2454, September 2011.

[10] Christophe Giraud, Sylvie Huet, and Nicolas Verzelen. High-dimensional regression with unknown variance. *submitted*, page arXiv:1109.5587v2 [math.ST].

[11] Z. Harchaoui and C. Lévy-Leduc. Multiple change-point estimation with a total variation penalty. *J. Amer. Statist. Assoc.*, 105(492):1480–1493, 2010.

[12] Zaïd Harchaoui and Céline Lévy-Leduc. Catching change-points with lasso. In John Platt, Daphne Koller, Yoram Singer, and Sam Roweis, editors, *NIPS*. Curran Associates, Inc., 2007.

[13] R. I. Hartley and A. Zisserman. *Multiple View Geometry in Computer Vision*. Cambridge University Press, June 2004.

[14] A. Iouditski, F. Kilinc Karzan, A. S. Nemirovski, and B. T. Polyak. On the accuracy of l1-filtering of signals with block-sparse structure. In *NIPS 24*, pages 1260–1268. 2011.

[15] S. Lambert-Lacroix and L. Zwald. Robust regression through the Huber's criterion and adaptive lasso penalty. *Electron. J. Stat.*, 5:1015–1053, 2011.

[16] David G. Lowe. Distinctive image features from scale-invariant keypoints. *International Journal of Computer Vision*, 60(2):91–110, 2004.

[17] E. Mammen and S. van de Geer. Locally adaptive regression splines. *Ann. Statist.*, 25(1):387–413, 1997.

[18] Nam H. Nguyen, Nasser M. Nasrabadi, and Trac D. Tran. Robust lasso with missing and grossly corrupted observations. In J. Shawe-Taylor, R.S. Zemel, P. Bartlett, F.C.N. Pereira, and K.Q. Weinberger, editors, *Advances in Neural Information Processing Systems 24*, pages 1881–1889. 2011.

[19] A. Rinaldo. Properties and refinements of the fused lasso. *Ann. Statist.*, 37(5B):2922–2952, 2009.

[20] Nicolas Städler, Peter Bühlmann, and Sara van de Geer. $\ell_1$-penalization for mixture regression models. *TEST*, 19(2):209–256, 2010.

[21] C. Strecha, W. von Hansen, L. Van Gool, P. Fua, and U. Thoennessen. On benchmarking camera calibration and multi-view stereo for high resolution imagery. In *Conference on Computer Vision and Pattern Recognition*, pages 1–8, 2009.

[22] J. F. Sturm. Using SeDuMi 1.02, a MATLAB toolbox for optimization over symmetric cones. *Optim. Methods Softw.*, 11/12(1-4):625–653, 1999.

[23] T. Sun and C.-H. Zhang. Comments on: $\ell_1$-penalization for mixture regression models. *TEST*, 19(2): 270–275, 2010.

[24] T. Sun and C.-H. Zhang. Scaled sparse linear regression. *arXiv:1104.4595*, 2011.

[25] R. Szeliski. *Computer Vision: Algorithms and Applications*. Texts in Computer Science. Springer, 2010.

[26] Robert Tibshirani. Regression shrinkage and selection via the lasso. *J. Roy. Statist. Soc. Ser. B*, 58(1): 267–288, 1996.

[27] Robert Tibshirani, Michael Saunders, Saharon Rosset, Ji Zhu, and Keith Knight. Sparsity and smoothness via the fused lasso. *J. R. Stat. Soc. Ser. B Stat. Methodol.*, 67(1):91–108, 2005.

[28] Sara A. van de Geer and Peter Bühlmann. On the conditions used to prove oracle results for the Lasso. *Electron. J. Stat.*, 3:1360–1392, 2009.

